# Some Solutions to the Missing Feature Problem in Vision

**Subutai Ahmad**
Siemens AG,
Central Research and Development
ZFE ST SN61, Otto-Hahn Ring 6
8000 München 83, Germany.
ahmad@icsi.berkeley.edu

**Volker Tresp**
Siemens AG,
Central Research and Development
ZFE ST SN41, Otto-Hahn Ring 6
8000 München 83, Germany.
tresp@inf21.zfe.siemens.de

## Abstract

In visual processing the ability to deal with missing and noisy information is crucial. Occlusions and unreliable feature detectors often lead to situations where little or no direct information about features is available. However the available information is usually sufficient to highly constrain the outputs. We discuss Bayesian techniques for extracting class probabilities given partial data. The optimal solution involves integrating over the missing dimensions weighted by the local probability densities. We show how to obtain closed-form approximations to the Bayesian solution using Gaussian basis function networks. The framework extends naturally to the case of noisy features. Simulations on a complex task (3D hand gesture recognition) validate the theory. When both integration and weighting by input densities are used, performance decreases gracefully with the number of missing or noisy features. Performance is substantially degraded if either step is omitted.

## 1 INTRODUCTION

The ability to deal with missing or noisy features is vital in vision. One is often faced with situations in which the full set of image features is not computable. In fact, in 3D object recognition, it is highly unlikely that all features will be available. This can be due to self-occlusion, occlusion from other objects, shadows, etc. To date the issue of missing features has not been dealt with in neural networks in a systematic way. Instead the usual practice is to substitute a single value for the missing feature (e.g. $0$, the mean value of the feature, or a pre-computed value) and use the network's output on that feature vector.

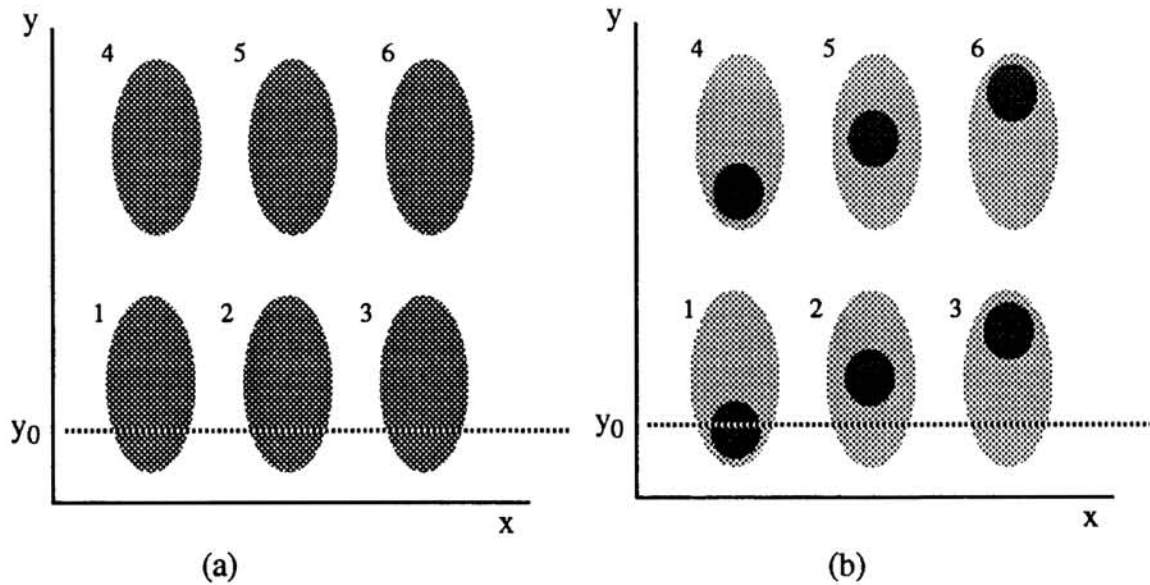

Figure 1. The images show two possible situations for a 6-class classification problem. (Dark shading denotes high-probability regions.) If the value of feature x is unknown, the correct solution depends both on the classification boundaries along the missing dimension and on the distribution of exemplars.

When the features are known to be noisy, the usual practice is to just use the measured noisy features directly. The point of this paper is to show that these approaches are not optimal and that it is possible to do much better.

A simple example serves to illustrate why one needs to be careful in dealing with missing features. Consider the situation depicted in Figure 1(a). It shows a *2-d* feature space with 6 possible classes. Assume a network has already been trained to correctly classify these regions. During classification of a novel exemplar, only feature y has been measured, as $y_0$; the value of feature $x$ is unknown. For each class $C_i$, we would like to compute $p(C_i|y)$. Since nothing is known about x, the classifier should assign equal probability to classes 1, 2, and 3, and zero probability to classes 4, 5, and 6. Note that substituting any *single* value will always produce the wrong result. For example, if the mean value of x is substituted, the classifier would assign a probability near 1 for class 2. To obtain the correct posterior probability, it is necessary to integrate the network output over all values of x. But there is one other fact to consider: the probability distribution over x may be highly constrained by the known value of feature y. With a distribution as in Figure 1(b) the classifier should assign class 1 the highest probability. Thus it is necessary to *integrate over x along the line $y=y_0$ weighted by the joint distribution p(x,y)*.

## 2  MISSING FEATURES

We first show how the intituitive arguments outlined above for missing inputs can be formalized using Bayes rule. Let $\hat{x}$ represent a complete feature vector. We assume the classifier outputs good estimates of $p(C_i|\hat{x})$ (most reasonable classifiers do - see (Richard & Lippmann, 1991)). In a given instance, $\hat{x}$ can be split up into $\hat{x}_c$, the vector of known (certain) features, and $\hat{x}_u$, the unknown features. When features are missing the task is to estimate $p(C_i|\hat{x}_c)$. Computing marginal probabilities we get:

$$p(C_i | \dot{x}_c) = \frac{p(C_i, \dot{x}_c)}{p(\dot{x}_c)} = \frac{\int p(C_i, \dot{x}_c, \dot{x}_u) \, d\dot{x}_u}{p(\dot{x}_c)} = \frac{\int p(C_i | \dot{x}_c, \dot{x}_u) \, p(\dot{x}_c, \dot{x}_u) \, d\dot{x}_u}{p(\dot{x}_c)} \qquad (1)$$

Note that $p(C_i | \dot{x}_c, \dot{x}_u)$ is approximated by the network output and that in order to use (1) effectively we need estimates of the joint probabilities of the inputs.

## 3 NOISY FEATURES

The missing feature scenario can be extended to deal with noisy inputs. (Missing features are simply noisy features in the limiting case of complete noise.) Let $\dot{x}_c$ be the vector of features measured with complete certainty, $\dot{x}_u$ the vector of measured, uncertain features, and $\dot{x}_{tu}$ the true values of the features in $\dot{x}_u$. $p(\dot{x}_u | \dot{x}_{tu})$ denotes our knowledge of the noise (i.e. the probability of measuring the (uncertain) value $\dot{x}_u$ given that the true value is $\dot{x}_{tu}$). We assume that this is independent of $\dot{x}_c$ and $C_i$, i.e. that $p(\dot{x}_u | \dot{x}_{tu}, \dot{x}_c, C_i) = p(\dot{x}_u | \dot{x}_{tu})$. (Of course the value of $\dot{x}_{tu}$ is dependent on $\dot{x}_c$ and $C_i$.) We want to compute $p(C_i | \dot{x}_c, \dot{x}_u)$. This can be expressed as:

$$p(C_i | \dot{x}_c, \dot{x}_u) = \frac{\int p(\dot{x}_c, \dot{x}_u, \dot{x}_{tu}, C_i) \, d\dot{x}_{tu}}{p(\dot{x}_c, \dot{x}_u)} \qquad (2)$$

Given the independence assumption, this becomes:

$$p(C_i | \dot{x}_c, \dot{x}_u) = \frac{\int p(C_i | \dot{x}_c, \dot{x}_{tu}) \, p(\dot{x}_c, \dot{x}_{tu}) \, p(\dot{x}_u | \dot{x}_{tu}) \, d\dot{x}_{tu}}{\int p(\dot{x}_c, \dot{x}_{tu}) \, p(\dot{x}_u | \dot{x}_{tu}) \, d\dot{x}_{tu}} \qquad (3)$$

As before, $p(C_i | \dot{x}_c, \dot{x}_{tu})$ is given by the classifier. (3) is almost the same as (1) except that the integral is also weighted by the noise model. Note that in the case of complete uncertainty about the features (i.e. the noise is uniform over the entire range of the features), the equations reduce to the missing feature case.

## 4 GAUSSIAN BASIS FUNCTION NETWORKS

The above discussion shows how to optimally deal with missing and noisy inputs in a Bayesian sense. We now show how these equations can be approximated using networks of Gaussian basis functions (GBF nets). Let us consider GBF networks where the Gaussians have diagonal covariance matrices (Nowlan, 1990). Such networks have proven to be useful in a number of real-world applications (e.g. Röscheisen *et al*, 1992). Each hidden unit is characterized by a mean vector $\dot{\mu}_j$ and by $\dot{\sigma}_j$, a vector representing the diagonal of the covariance matrix. The network output is:

$$y_i(\dot{x}) = \frac{\sum_j w_{ij} b_j(\dot{x})}{\sum_j b_j(\dot{x})}$$

$$\text{with } b_j(\dot{x}) = \pi_j n(\dot{x};\vec{\mu}_j,\vec{\sigma}_j^2) = \frac{\pi_j}{(2\pi)^{\frac{d}{2}} \prod_k \vec{\sigma}_{kj}} exp\left[-\sum_i \frac{(x_i - \mu_{ji})^2}{2\vec{\sigma}_{ji}^2}\right] \quad (4)$$

$w_{ji}$ is the weight from the j'th basis unit to the i'th output unit, $\pi_j$ is the probability of choosing unit j, and d is the dimensionality of $\dot{x}$.

## 4.1 GBF NETWORKS AND MISSING FEATURES

Under certain training regimes such as Gaussian mixture modeling, EM or "soft cluster-ing" (Duda & Hart, 1973; Dempster *et al*, 1977; Nowlan, 1990) or an approximation as in (Moody & Darken, 1988) the hidden units adapt to represent local probability densities. In particular $y_i(\dot{x}) \approx p(C_i|\dot{x})$ and $p(\dot{x}) \approx \sum_j b_j(\dot{x})$. This is a major advantage of this archi-tectur and can be exploited to obtain closed form solutions to (1) and (3). Substituting into (3) we get:

$$p(C_i|\dot{x}_c,\dot{x}_u) \approx \frac{\int (\sum_j w_{ij}b_j(\dot{x}_c,\dot{x}_{tu})) p(\dot{x}_u|\dot{x}_{tu}) d\dot{x}_{tu}}{\int (\sum_j b_j(\dot{x}_c,\dot{x}_{tu})) p(\dot{x}_u|\dot{x}_{tu}) d\dot{x}_{tu}} \quad (5)$$

For the case of missing features equation (5) can be computed directly. As noted before, equation (1) is simply (3) with $p(\dot{x}_u|\dot{x}_{tu})$ uniform. Since the infinite integral along each dimension of a multivariate normal density is equal to one we get:

$$p(C_i|\dot{x}_c) \approx \frac{\sum_j w_{ji}b_j(\dot{x}_c)}{\sum_j b_j(\dot{x}_c)} \quad (6)$$

(Here $b_j(\dot{x}_c)$ denotes the same function as in   except that it is only evaluated over the known dimensions given by $\dot{x}_c$.) Equation (6) is appealing since it gives us a simple closed form solution. Intuitively, the solution is nothing more than projecting the Gaussians onto the dimensions which are available and evaluating the resulting network. As the number of training patterns increases, (6) will approach the optimal Bayes solution.

## 4.2 GBF NETWORKS AND NOISY FEATURES

With noisy features the situation is a little more complicated and the solution depends on the form of the noise. If the noise is known to be uniform in some region $[\dot{a},\dot{b}]$ then equation (5) becomes:

$$p(C_i|\dot{x}_c,\dot{x}_u) \approx \frac{\sum_j w_{ij}b_j(\dot{x}_c) \prod_{i \in U} [N(b_i;\mu_{ij},\sigma_{ij}^2) - N(a_i;\mu_{ij},\sigma_{ij}^2)]}{\sum_j b_j(\dot{x}_c) \prod_{i \in U} [N(b_i;\mu_{ij},\sigma_{ij}^2) - N(a_i;\mu_{ij},\sigma_{ij}^2)]} \quad (7)$$

Here $\vec{\mu}_{ij}$ and $\vec{\sigma}_{ij}^2$ select the i'th component of the j'th mean and variance vectors. U ranges over the noisy feature indices. Good closed form approximations to the normal distribution function $N(x;\mu,\sigma^2)$ are available (Press et al, 1986) so (7) is efficiently computable.

With zero-mean Gaussian noise with variance $\sigma_u^2$, we can also write down a closed form solution. In this case we have to integrate a product of two Gaussians and end up with:

$$p(C_i|\vec{x}_c,\vec{x}_u) = \frac{\sum_j w_{ij}b'_j(\vec{x}_c,\vec{x}_u)}{\sum_j b'_j(\vec{x}_c,\vec{x}_u)} \text{ with } b'_j(\vec{x}_c,\vec{x}_u) = n(\vec{x}_u;\vec{\mu}_{ju},\vec{\sigma}_u^2+\vec{\sigma}_{ju}^2)b_j(\vec{x}_c).$$

## 5 BACKPROPAGATION NETWORKS

With a large training set, the outputs of a sufficiently large network trained with back-propagation converges to the optimal Bayes *a posteriori* estimates (Richard & Lippmann, 1992). If $B_i(\vec{x})$ is the output of the i'th output unit when presented with input $\vec{x}$, $B_i(\vec{x}) \approx p(C_i|\vec{x})$. Unfortunately, access to the input distribution is not available with back-propagation. Without prior knowledge it is reasonable to assume a uniform input distribution, in which case the right hand side of (3) simplifies to:

$$p(C_i|\vec{x}_c) \approx \frac{\int p(C_i|\vec{x}_c,\vec{x}_{tu})p(\vec{x}_u|\vec{x}_{tu})d\vec{x}_{tu}}{\int p(\vec{x}_u|\vec{x}_{tu})d\vec{x}_{tu}} \tag{8}$$

The integral can be approximated using standard Monte Carlo techniques. With uniform noise in the interval $[\vec{a},\vec{b}]$, this becomes (ignoring normalizing constants):

$$p(C_i|\vec{x}_c) \approx \int_{\vec{a}}^{\vec{b}} B_i(\vec{x}_c,\vec{x}_{tu})d\vec{x}_{tu} \tag{9}$$

With missing features the integral in (9) is computed over the entire range of each feature.

## 6 AN EXAMPLE TASK: 3D HAND GESTURE RECOGNITION

A simple realistic example serves to illustrate the utility of the above techniques. We consider the task of recognizing a set of hand gestures from single 2D images independent of 3D orientation (Figure 2). As input, each classifier is given the 2D polar coordinates of the five fingertip positions relative to the 2D center of mass of the hand (so the input space is 10-dimensional). Each classifier is trained on a training set of 4368 examples (624 poses for each gesture) and tested on a similar independent test set.

The task forms a good benchmark for testing performance with missing and uncertain inputs. The classification task itself is non-trivial. The classifier must learn to deal with hands (which are complex non-rigid objects) and with perspective projection (which is non-linear and non-invertible). In fact it is impossible to obtain a perfect score since in certain poses some of the gestures are indistinguishable (e.g. when the hand is pointing directly at the screen). Moreover, the task is characteristic of real vision problems. The

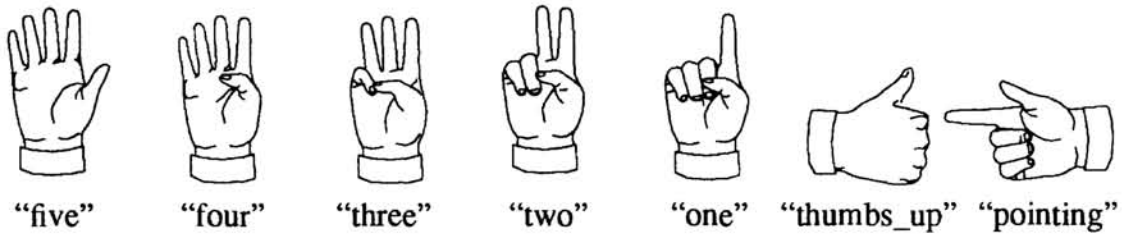

"five"     "four"     "three"     "two"     "one"    "thumbs_up"  "pointing"

Figure 2. Examples of the 7 gestures used to train the classifier. A 3D computer model of the hand is used to generate images of the hand in various poses. For each training example, we choose a 3D orientation, compute the 3D positions of the fingertips and project them onto 2D. For this task we assume that the correspondence between image and model features are known, and that during training all feature values are always available.

position of each finger is highly (but not completely) constrained by the others resulting in a very non-uniform input distribution. Finally it is often easy to see what the classifier should output if features are uncertain. For example suppose the real gesture is "five" but for some reason the features from the thumb are not reliably computed. In this case the gestures "four" and "five" should both get a positive probability whereas the rest should get zero. In many such cases only a single class should get the highest score, e.g. if the features for the little finger are uncertain the correct class is still "five".

We tried three classifiers on this task: standard sigmoidal networks trained with backpropagation (BP), and two types of gaussian networks as described in . In the first (Gauss-RBF), the gaussians were radial and the centers were determined using k-means clustering as in (Moody & Darken, 1988). $\sigma^2$ was set to twice the average distance of each point to its nearest gaussian (all gaussians had the same width). After clustering, $\pi_j$ was set to

$$\sum_k \left[ \frac{n(\vec{x}_k; \vec{\mu}_j; \vec{\sigma}_j^2)}{\sum_i n(\vec{x}_k; \vec{\mu}_i; \vec{\sigma}_i^2)} \right].$$ The output weights were then determined using LMS gradient

descent. In the second (Gauss-G), each gaussian had a unique diagonal covariance matrix. The centers and variances were determined using gradient descent on all the parameters (Röscheisen *et al*, 1992). Note that with this type of training, even though gaussian hidden units are used, there is no guarantee that the distribution information will be preserved.

All classifiers were able to achieve a reasonable performance level. BP with 60 hidden units managed to score 95.3% and 93.3% on the training and test sets, respectively. Gauss-G with 28 hidden units scored 94% and 92%. Gauss-RBF scored 97.7% and 91.4% and required 2000 units to achieve it. (Larger numbers of hidden units led to overfitting.) For comparison, nearest neighbor achieves a score of 82.4% on the test set.

## 6.1 PERFORMANCE WITH MISSING FEATURES

We tested the performance of each network in the presence of missing features. For backpropagation we used a numerical approximation to equation (9). For both gaussian basis function networks we used equation (6). To test the networks we randomly picked samples from the test set and deleted random features. We calculated a performance score as the percentage of samples where the correct class was ranked as one of the top two classes. Figure 3 displays the results. For comparison we also tested each classifier by substituting the mean value of each missing feature and using the normal update equation.

As predicted by the theory the performance of Gauss-RBF using (6) was consistently better than the others. The fact that BP and Gauss-G performed poorly indicates that the distribution of the features must be taken into account. The fact that using the mean value is

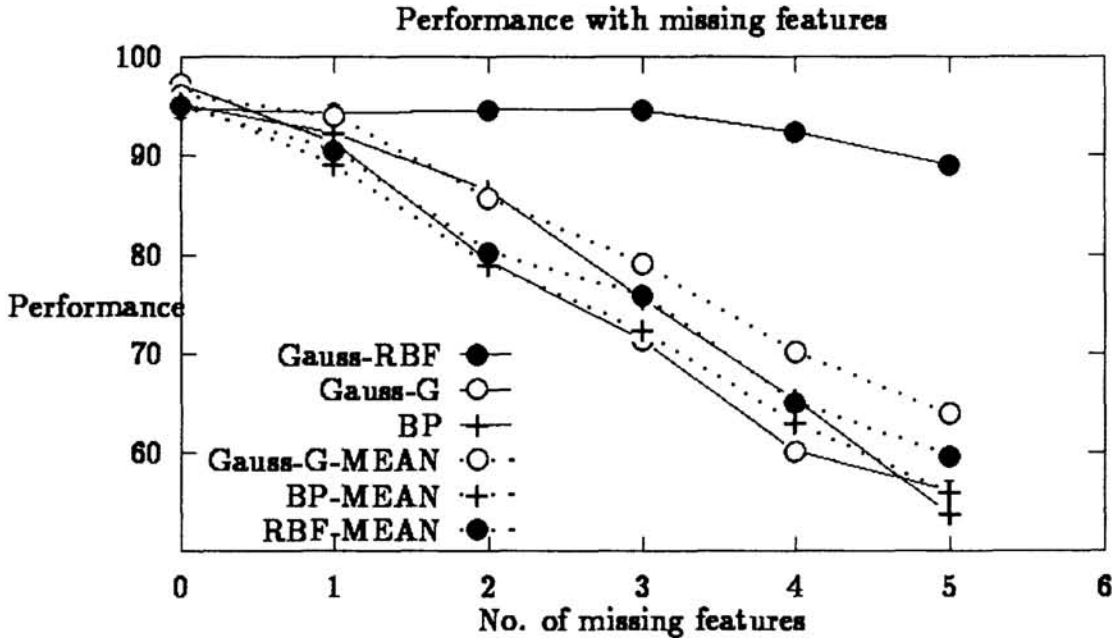

Figure 3. The performance of various classifiers when dealing with missing features. Each data point denotes an average over 1000 random samples from an independent test set. For each sample, random features were considered missing. Each graph plots the percentage of samples where the correct class was one of the top two classes.

insufficient indicates that the integration step must also be carried out. Perhaps most encouraging is the result that even with 50% of the features missing, Gauss-RBF ranks the correct class among the top two 90% of the time. This clearly shows that a significant amount of information can be extracted even with a large number of missing features.

## 6.2 PERFORMANCE WITH NOISY FEATURES

We also tested the performance of each network in the presence of noisy features. We randomly picked samples from the test set and added uniform noise to random features. The noise interval was calculated as $[x_i - 2\sigma_i, x_i + 2\sigma_i]$ where $x_i$ is the feature value and $\sigma_i$ is the standard deviation of that feature over the training set. For BP we used equation (9) and for the GBF networks we used equation (7). Figure 3 displays the results. For comparison we also tested each classifier by substituting the noisy value of each noisy feature and using the normal update equation (RBF-N, BP-N, and Gauss-GN). As with missing features, the performance of Gauss-RBF was significantly better than the others when a large number of features were noisy.

## 7 DISCUSSION

The results demonstrate the advantages of estimating the input distribution and integrating over the missing dimensions, at least on this task. They also show that good classification performance alone does not guarantee good missing feature performance. (Both BP and Gauss-G performed better than Gauss-RBF on the test set.) To get the best of both worlds one could use a hybrid technique utilizing separate density estimators and classifiers although this would probably require equations (1) and (3) to be numerically integrated.

One way to improve the performance of BP and Gauss-G might be to use a training set that contained missing features. Given the unusual distributions that arise in vision, in order to guarantee accuracy such a training set should include every possible combination

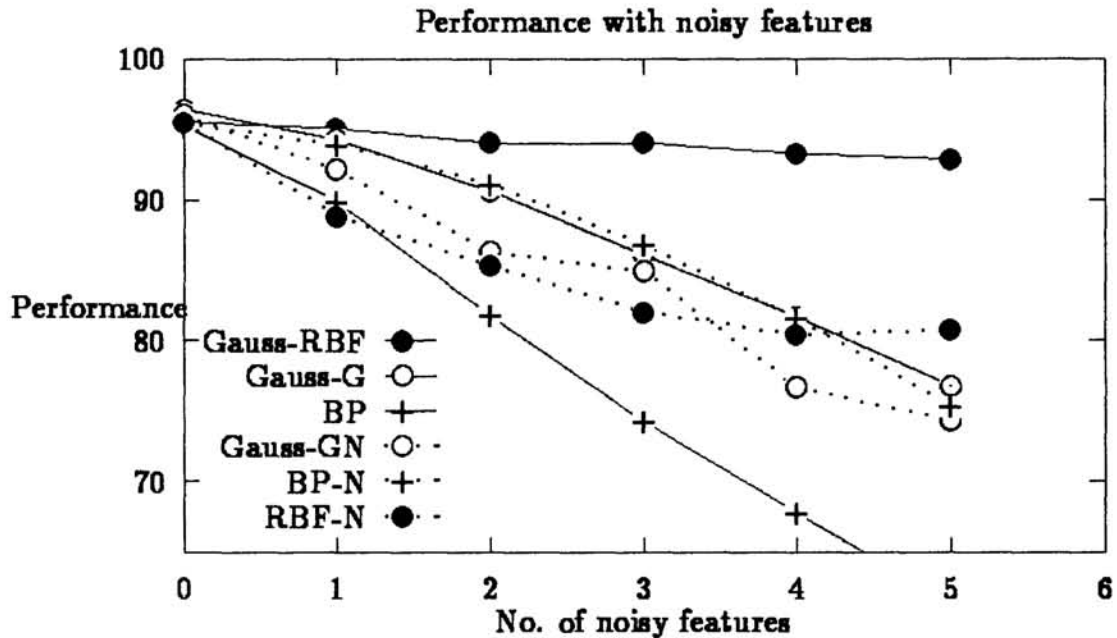

Figure 4. As in Figure 3 except that the performance with noisy features is plotted.

of missing features. In addition, for each such combination, enough patterns must be included to accurately estimate the posterior density. In general this type of training is intractable since the number of combinations is exponential in the number of features. Note that if the input distribution is available (as in Gauss-RBF), then such a training scenario is unnecessary.

## Acknowledgements

We thank D. Goryn, C. Maggioni, S. Omohundro, A. Stolcke, and R. Schuster for helpful discussions, and especially B. Wirtz for providing the computer hand model. V.T. is supported in part by a grant from the Bundesministerium für Forschung und Technologie.

## References

A.P. Dempster, N.M. Laird, and D.B. Rubin. (1977) Maximum-likelihood from incomplete data via the EM algorithm. *J. Royal Statistical Soc. Ser. B*, 39:1-38.

R.O. Duda and P.E. Hart. (1973) *Pattern Classification and Scene Analysis*. John Wiley & Sons, New York.

J. Moody and C. Darken. (1988) Learning with localized receptive fields. In: D. Touretzky, G. Hinton, T. Sejnowski, editors, *Proceedings of the 1988 Connectionist Models Summer School*, Morgan Kaufmann, CA.

S. Nowlan. (1990) Maximum Likelihood Competitive Learning. In: *Advances in Neural Information Processing Systems 4*, pages 574-582.

W.H. Press, B.P. Flannery, S.A. Teukolsky, and W.T. Vetterling. (1986) *Numerical Recipes: The Art of Scientific Computing*, Cambridge University Press, Cambridge, UK.

M. D. Richard and R.P. Lippmann. (1991) Neural Network Classifiers Estimate Bayesian *a posteriori* Probabilities, *Neural Computation*, 3:461-483.

M. Röscheisen, R. Hofman, and V. Tresp. (1992) Neural Control for Rolling Mills: Incorporating Domain Theories to Overcome Data Deficiency. In: *Advances in Neural Information Processing Systems 4*, pages 659-666.